# Best-First Model Merging for
# Dynamic Learning and Recognition

**Stephen M. Omohundro**
International Computer Science Institute
1947 Center Street, Suite 600
Berkeley, California 94704

## Abstract

"Best-first model merging" is a general technique for dynamically choosing the structure of a neural or related architecture while avoiding overfitting. It is applicable to both learning and recognition tasks and often generalizes significantly better than fixed structures. We demonstrate the approach applied to the tasks of choosing radial basis functions for function learning, choosing local affine models for curve and constraint surface modelling, and choosing the structure of a balltree or bumptree to maximize efficiency of access.

## 1 TOWARD MORE COGNITIVE LEARNING

Standard backpropagation neural networks learn in a way which appears to be quite different from human learning. Viewed as a cognitive system, a standard network always maintains a complete model of its domain. This model is mostly wrong initially, but gets gradually better and better as data appears. The net deals with all data in much the same way and has no representation for the strength of evidence behind a certain conclusion. The network architecture is usually chosen before any data is seen and the processing is much the same in the early phases of learning as in the late phases.

Human and animal learning appears to proceed in quite a different manner. When an organism has not had many experiences in a domain of importance to it, each individual experience is critical. Rather than use such an experience to slightly modify the parameters of a global model, a better strategy is to remember the experience in detail. Early in learning, an organism doesn't know which features of an experience are important unless it has a strong prior knowledge of the domain. Without such prior knowledge, its best strategy is to generalize on the basis of a similarity measure to individual stored experiences. (Shepard, 1987) shows that there is a universal exponentially decaying form for this kind of similarity based generalization over a wide variety of sensory domains in several studied species. As experiences accumulate, the organism eventually gets enough data to reliably validate models from complex classes. At this point the animal need no longer remember individual experiences, but rather only the discovered generalities (eg. as rules). With such a strategy, it is possible for a system to maintain a measure of confidence in it its predictions while building ever more complex models of its environment.

Systems based on these two types of learning have also appeared in the neural network, statistics and machine learning communities. In the learning literature one finds both "table-lookup" or "memory-based" methods and "parameter-fitting" methods. In statistics the distinction is made between "non-parametric" and "parametric" methods. Table-lookup methods work by storing examples and generalize to new situations on the basis of similarity to the old ones. Such methods are capable of one-shot learning and have a measure of the applicability of their knowledge to new situations but are limited in their generalization capability. Parameter fitting models choose the parameters of a predetermined model to best fit a set of examples. They usually take longer to train and are susceptible to computational difficulties such as local maxima but can potentially generalize better by extending the influence of examples over the whole space. Aside from computational difficulties, their fundamental problem is overfitting, ie. having insufficient data to validate a particular parameter setting as useful for generalization.

## 2  OVERFITTING IN LEARNING AND RECOGNITION

There have been many recent results (eg. based on the Vapnik-Chervonenkis dimension) which identify the number of examples needed to validate choices made from specific parametric model families. We would like a learning system to be able to induce extremely complex models of the world but we don't want to have to present it with the enormous amount of data needed to validate such a model unless it is really needed. (Vapnik, 1982) proposes a technique for avoiding overfitting while allowing models of arbitrary complexity. The idea is to start with a nested familty of model spaces, whose members contain ever more complex models. When the system has only a small amount of data it can only validate models in in the smaller model classes. As more data arrives, however, the more complex classes may be considered. If at any point a fit is found to within desired tolerances, however, only the amount of data needed by the smallest class containing the chosen model is needed. Thus there is the potential for choosing complex models without penalizing situations in which the model is simple. The model merging approach may be viewed in these terms except that instead of a single nested family, there is a widely branching tree of model spaces.

Like learning, recognition processes (visual, auditory, etc.) aim at constructing models from data. As such they are subject to the same considerations regarding overfitting. Figure 1 shows a perceptual example where a simpler model (a single segment) is perceptually chosen to explain the data (4 almost collinear dots) than a more complex model (two segments) which fits the data better. An intuitive explanations is that if the dots were generated by two segments, it would be an amazing coincidence that they are almost collinear, if it were generated by one, that fact is easily explained. Many of the Gestalt phenomena can be

considered in the same terms. Many of the processes used in recognition (eg. segmentation, grouping) have direct analogs in learning and vice versa.

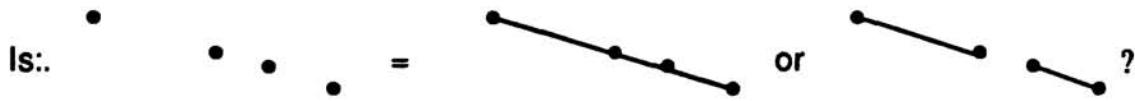

Figure 1: An example of Occam's razor in recognition.

There has been much recent interest in the network community in Bayesian methods for model selection while avoiding overfitting (eg. Buntine and Weigend, 1992 and MacKay 1992). Learning and recognition fit naturally together in a Bayesian framework. The Bayesian approach makes explicit the need for a prior distribution. The posterior distribution generated by learning becomes the prior distribution for recognition. The model merging process described in this paper is applicable to both phases and the knowledge representation it suggests may be used for both processes as well.

There are at least three properties of the world that may be encoded in a prior distribution and have a dramatic effect on learning and recognition and are essential to the model merging approach. The *continuity prior* is that the world is geometric and unless there is contrary data a system should prefer continuous models over discontinuous ones. This prior leads to a wide variety of what may be called "geometric learning algorithms" (Omohundro, 1990). The *sparseness prior* is that the world is sparsely interacting. This says that probable models naturally decompose into components which only directly affect one another in a sparse manner. The primary origin of this prior is that physical objects usually only directly affect nearby objects in space and time. This prior is responsible for the success of representations such as Markov random fields and Bayesian networks which encode conditional independence relations. Even if the individual models consist of sparsely interacting components, it still might be that the data we receive for learning or recognition depends in an intricate way on all components. The *locality prior* prefers models in which the data decomposes into components which are directly affected by only a small number of model components. For example, in the learning setting only a small portion of the knowledge base will be relevant to any specific situation. In the recognition setting, an individual pixel is determined by only a small number of objects in the scene. In geometric settings, a localized representation allows only a small number of model parameters to affect any individual prediction.

## 3 MODEL MERGING

Based on the above considerations, an ideal learning or recognition system should model the world using a collection of sparsely connected, smoothly parameterized, localized models. This is an apt description of many of the neural network models currently in use. Bayesian methods provide an optimal means for induction with such a choice of prior over models but are computationally intractable in complex situations. We would therefore like to develop heuristic approaches which approximate the Bayesian solution and avoid overfitting. Based on the idealization of animal learning in the first section, we would like is a system which smoothly moves between a memory-based regime in which the models are the data into ever more complex parameterized models. Because of the locality prior, model

components only affect a subset of the data. We can therefore choose the complexity of components which are relevant to different portions of the data space according to the data which has been received there. This allows for reliably validated models of extremely high complexity in some regions of the space while other portions are modeled with low complexity. If only a small number of examples have been seen in some region, these are simply remembered and generalization is based on similarity. As more data arrives, if regularities are found and there is enough data present to justify them, more complex parameterized models are incorporated.

There are many possible approaches to implementing such a strategy. We have investigated a particular heuristic which can be made computationally efficient and appears to work well in a variety of areas. The *best-first model merging* approach is applicable in a variety of situations in which complex models are constructed by combining simple ones. The idea is to improve a complex model by replacing two of its component models by a single model. This "merged" model may be in the same family as the original components. More interestingly, because the combined data from the merged components is used in determining the parameters of the merged model, it may come from a larger parameterized class. The critical idea is to never allow the system to hypothesize a model which is more complex than can be justified by the data it is based on. The "best-first" aspect is to always choose to merge the pair of models which decrease the likelihood of the data the least. The merging may be stopped according to a variety of criteria which are now applied to individual model components rather than the entire model. Examples of such criteria are those based on cross-validation, Bayesian Occam factors, VC bounds, etc. In experiments in a variety of domains, this approach does an excellent job of discovering regularities and allocating modelling resources efficiently.

## 3 MODEL MERGING VS. K-MEANS FOR RBF'S

Our first example is the problem of choosing centers in radial basis function networks for approximating functions. In the simplest approach, a radial basis function (eg. a Gaussian) is located at each training input location. The induced function is a linear combination of these basis functions which minimizes the mean square error of the training examples. Better models may be obtained by using fewer basis functions than data points. Most work on choosing the centers of these functions uses a clustering technique such as k-means (eg. Moody and Darken, 1989). This is reasonable because it puts the representational power of the model in the regions of highest density where errors are more critical. It ignores the structure of the modelled function, however. The model merging approach starts with a basis function at each training point and successively merges pairs which increase the training error the least. We compared this approach with the k-means approach in a variety of circumstances.

Figure 2 shows an example where the function on the plane to be learned is a sigmoid in $x$ centered at 0 and is constant in $y$. Thus the function varies most along the $y$ axis. The data is drawn from a Gaussian distribution which is centered at (-.5,0). 21 training samples were drawn from this distribution and from these a radial basis function network with 6 Gaussian basis functions was learned. The X's in the figure show the centers chosen by k-means. As expected, they are clustered near the center fo the Gaussian source distribution. The triangles show the centers chosen by best-first model merging. While there is some tendency to focus on the source center, there is also a tendency to represent the region where the modelled function varies the most. The training error is over 10 times less with model merging

and the test error on an independent test set is about 3 times lower. These results were typical in variety of test runs. This simple example shows one way in which underlying structure is naturally discovered by the merging technique.

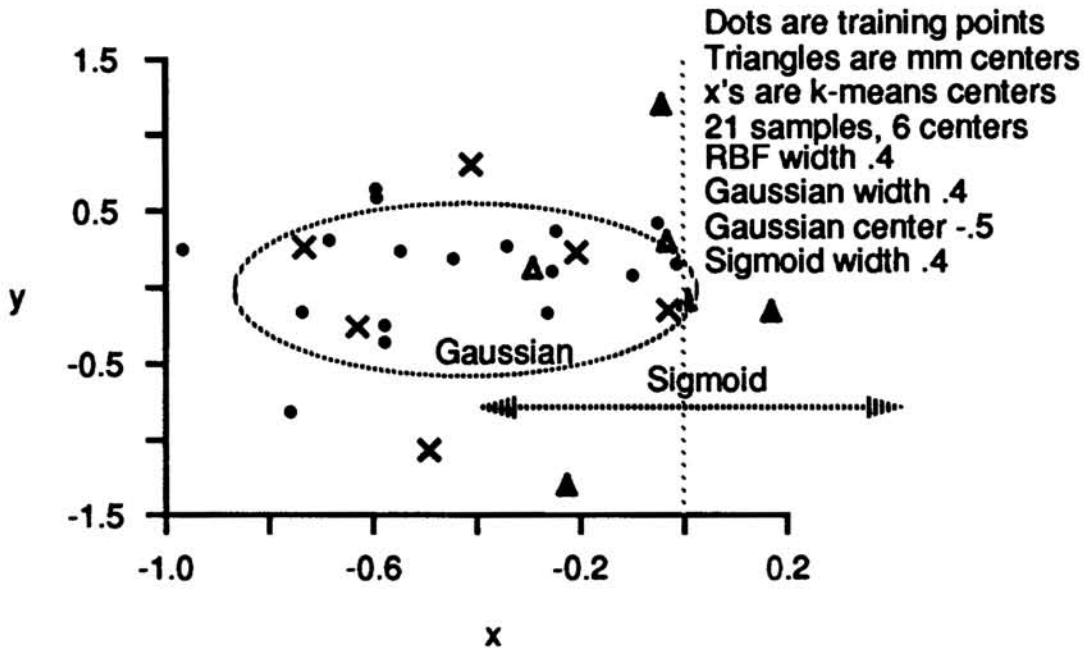

Figure 2: Radial basis function centers in two dimensions chosen by model merging and by k-means. The dots show the 21 training samples. The x's are the centers chosen by k-means, the triangles by model merging. The training error was .008098 for k-means and .000604 for model merging. The test error was .012463 for k-means and .004638 for model merging.

## 4  APPROXIMATING CURVES AND SURFACES

As a second intuitive example, consider the problem of modelling a curve in the plane by a combination of straight line segments. The error function may be taken as the mean square error over each curve point to the nearest segment point. A merging step in this case consists of replacing two segments by a single segment. We always choose that pair such that the merged segment increases the error the least. Figure 3 shows the approximations generated by this strategy. It does an excellent job at identifying the essentially linear portions of the curve and puts the boundaries between component models at the "corners". The corresponding "top-down" approach would start with a single segment and repeatedly split it. This approach sometimes has to make decisions too early and often misses the corners in the curve. While not shown in the figure, as repeated mergings take place, more data is available for each segment. This would allow us to use more complex models than linear segments such as Bezier curves. It is possible to reliably induce a representation which is linear in some portions and higher order in others. Such models potentially have many parameters and would be subject to overfitting if they were learned directly rather than by going through merge steps.

Exactly the same strategy may be applied to modelling higher-dimensional constraint surfaces by hyperplanes or functions by piecewise linear portions. The model merging ap-

proach naturally complements the efficient mapping and constraint surface representations described in (Omohundro, 1991) based on bumptrees.

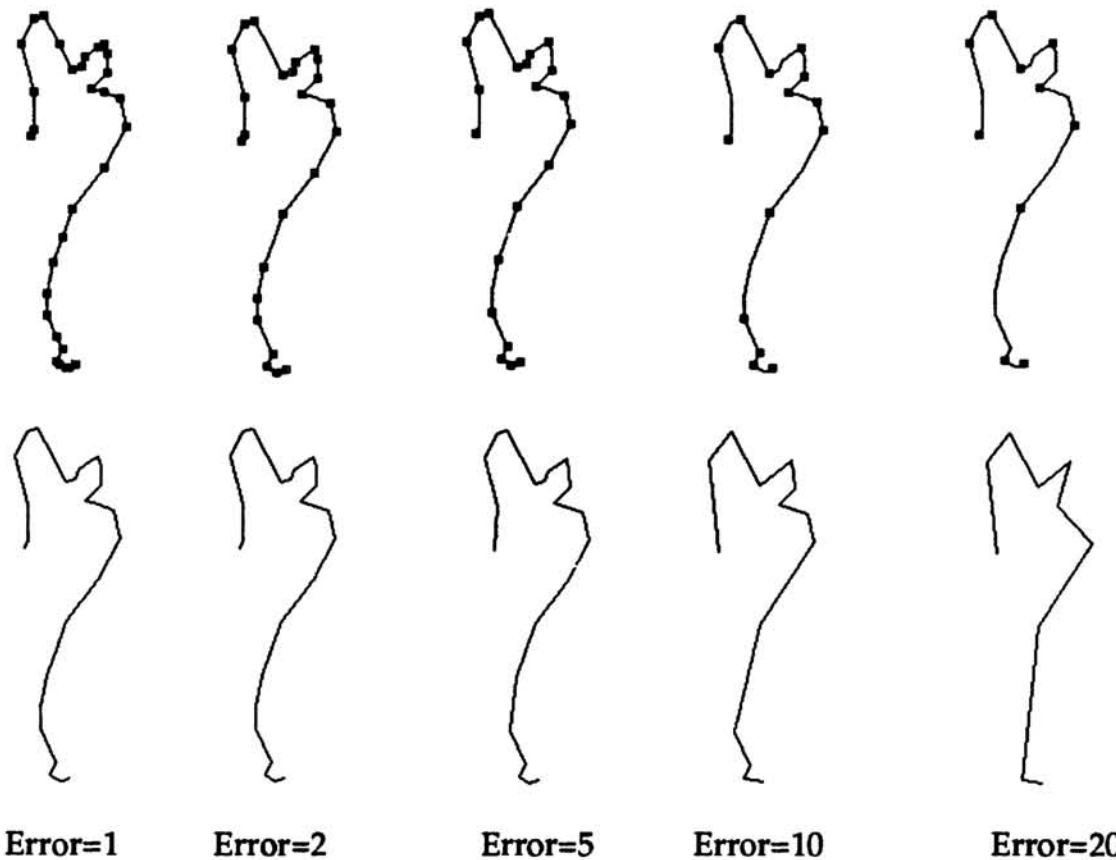

Error=1          Error=2          Error=5          Error=10          Error=20

Figure 3: Approximation of a curve by best-first merging of segment models. The top row shows the endpoints chosen by the algorithm at various levels of allowed error. The bottom row shows the corresponding approximation to the curve.

Notice, in this example, that we need only consider merging neighboring segments as the increased error in merging non-adjoinging segments would be too great. This imposes a locality on the problem which allows for extremely efficient computation. The idea is to maintain a priority queue with all potential merges on it ordered by the increase in error caused by the merge. This consists of only the neighboring pairs (of which there are n-1 if there are n segments). The top pair on the queue is removed and the merge operation it represents is performed if it doesn't violate the stopping critera. The other potential merge pairs which incorporated the merged segments must be removed from the queue and the new possible mergings with the generated segment must be inserted (alternatively, nothing need be removed and each pair is checked for viability when it reaches the top of the queue). The neighborhood structure allows each of the operations to be performed quickly with the appropriate data structures and the entire merging process takes a time which is linear (or linear times logarithmic) in the number of component models. Complex time-varying curves may easily be processed in real time on typical workstations. In higher dimensions, hierarchical geometric data structures (as in Omohundro, 1987, 1990) allow a similar reduction in computation based on locality.

## 5 BALLTREE CONSTRUCTION

The model merging approach is applicable to a wide variety of adaptive structures. The "balltree" structure described in (Omohundro, 1989) provides efficient access to regions in geometric spaces. It consists of a nested hierarchy of hyper-balls surrounding given leaf balls and efficiently supports querries which test for intersection, inclusion, or nearness to a leaf ball. The balltree construction algorithm itself provides an example of a best-first merge approach in a higher dimensional space. To determine the best hierarchy we can merge the leaf balls pairwise in such a way that the total volume of all the merged regions is as small as possible. The figure compares the quality of balltrees constructed using best-first merging to those constructed using top-down and incremental algorithms. As in other domains, the top-down approach has to make major decisions too early and often makes suboptimal choices. The merging approach only makes global decisions after many local decisions which adapt well to the structure.

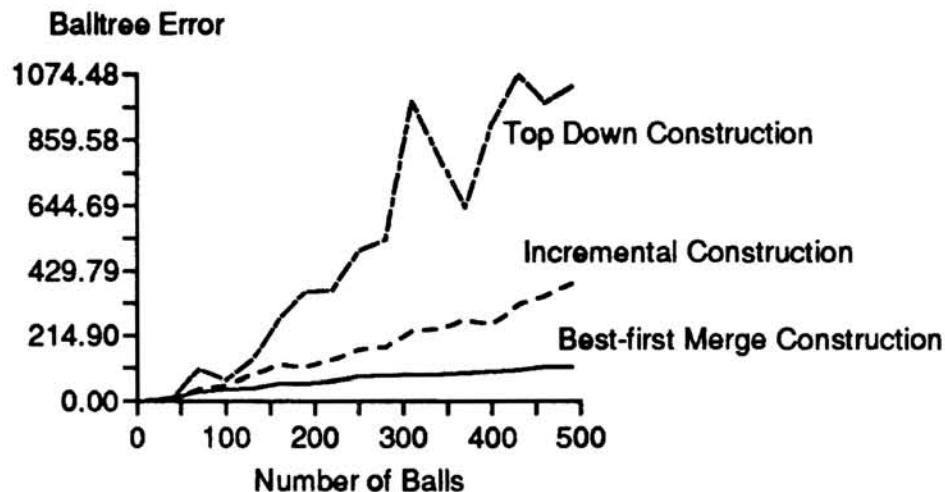

Figure 4: Balltree error as a function of number of balls for the top-down, incremental, and best-first merging construction methods. Leaf balls have uniformly distributed centers in 5 dimensions with radii uniformly distributed less than .1.

## 6 CONCLUSION

We have described a simple but powerful heuristic for dynamically building models for both learning and recognition which constructs complex models that adapt well to the underlying structure. We presented three different examples which only begin to touch on the possibilities. To hint at the broad applicability, we will briefly describe several other applications we are currently examining.

In (Omohundro, 1991) we presented an efficient structure for modelling mappings based on a collection of local mapping models which were combined according to a partition of unity formed by "influence functions" associated with each model. This representation is very flexible and can be made computationally efficient. While in the experiments of that paper, the local models were affine functions (constant plus linear), they may be chosen from any desired class. The model merging approach builds such a mapping representation by successively merging models and replacing them with a new model whose influence

function extends over the range of the two original influence functions. Because it is based on more data, the new model can be chosen from a larger complexity class of functions than the originals.

One of the most fundamental inductive tasks is density estimation, ie. estimating a probablity distribution from samples drawn from it. A powerful standard technique is adaptvie kernel estimation in which a normalized Gaussian (or other kernel) is placed at each sample point with a width determined by the local sample density (Devroye and Gyorfi, 1985). Model merging can be applied to improve the generalization performance of this approach by choosing successively more complex component densities once enough data has accumulated by merging. For example, consider a density supported on a curve in a high dimensional space. Initially the estimate will consist of radially-symmetric Gaussians at each sample point. After successive mergings, however, the one-dimensional linear structure can be discovered (and the Gaussian components be chosen from the larger class of extended Gaussians) and the generalization dramatically improved.

Other natural areas of application include inducing the structure of hidden Markov models, stochastic context-free grammars, Markov random fields, and Bayesian networks.

## References

D. H. Ballard and C. M. Brown. *(1982) Computer Vision*. Englewood Cliffs, N. J: Prentice-Hall.

W. L. Buntine and A. S. Weigend. (1992) Bayesian Back-Propagation. To appear in: *Complex Systems*.

L. Devroye and L. Gyorfi. (1985) *Nonparametric Density Estimation: The L1 View*, New York: Wiley.

D. J. MacKay. (1992) A Practical Bayesian Framework for Backprop Networks. Caltech preprint.

J. Moody and C. Darken. (1989) Fast learning in networks of locally-tuned processing units. *Neural Computation*, 1, 281-294.

S. M. Omohundro. (1987) Efficient algorithms with neural network behavior. *Complex Systems* 1:273-347.

S. M. Omohundro. (1989) Five balltree construction algorithms. *International Computer Science Institute Technical Report* TR-89-063.

S. M. Omohundro. (1990) Geometric learning algorithms. *Physica D* 42:307-321.

S. M. Omohundro. (1991) Bumptrees for Efficient Function, Constraint, and Classification Learning. In Lippmann, Moody, and Touretzky, (eds.) *Advances in Neural Information Processing Systems 3*. San Mateo, CA: Morgan Kaufmann Publishers.

R. N. Shepard. (1987) Toward a universal law of generalization for psychological science. *Science*.

V. Vapnik. (1982) *Estimation of Dependences Based on Empirical Data*, New York: Springer-Verlag.
